# Adaptive Soft Weight Tying using Gaussian Mixtures

**Steven J. Nowlan**
Computational Neuroscience Laboratory
The Salk Institute, P.O. Box 5800
San Diego, CA 92186-5800

**Geoffrey E. Hinton**
Department of Computer Science
University of Toronto
Toronto, Canada M5S 1A4

## Abstract

One way of simplifying neural networks so they generalize better is to add an extra term to the error function that will penalize complexity. We propose a new penalty term in which the distribution of weight values is modelled as a mixture of multiple gaussians. Under this model, a set of weights is simple if the weights can be clustered into subsets so that the weights in each cluster have similar values. We allow the parameters of the mixture model to adapt at the same time as the network learns. Simulations demonstrate that this complexity term is more effective than previous complexity terms.

## 1  Introduction

A major problem in training artificial neural networks is to ensure that they will generalize well to cases that they have not been trained on. Some recent theoretical results (Baum and Haussler, 1989) have suggested that in order to guarantee good generalization the amount of information required to directly specify the output vectors of all the training cases must be considerably larger than the number of independent weights in the network. In many practical problems there is only a small amount of labelled data available for training and this creates problems for any approach that uses a large, homogeneous network with many independent weights. As a result, there has been much recent interest in techniques that can train large networks with relatively small amounts of labelled data and still provide good generalization performance.

In order to improve generalization, the number of free parameters in the network must be reduced. One of the oldest and simplest approaches to removing excess degrees of freedom from a network is to add an extra term to the error function

that penalizes complexity:

$$\text{cost} = \text{data-misfit} + \lambda\,\text{complexity} \tag{1}$$

During learning, the network is trying to find a locally optimal trade-off between the data-misfit (the usual error term) and the complexity of the net. The relative importance of these two terms can be estimated by finding the value of $\lambda$ that optimizes generalization to a validation set. Probably the simplest approximation to complexity is the sum of the squares of the weights, $\sum_i w_i^2$. Differentiating this complexity measure leads to simple *weight decay* (Plaut, Nowlan and Hinton, 1986) in which each weight decays towards zero at a rate that is proportional to its magnitude. This decay is countered by the gradient of the error term, so weights which are not critical to network performance, and hence always have small error gradients, decay away leaving only the weights necessary to solve the problem.

The use of a $\sum_i w_i^2$ penalty term can also be interpreted from a Bayesian perspective.[1] The "complexity" of a set of weights, $\lambda \sum_i w_i^2$, may be described as its negative log probability density under a radially symmetric gaussian prior distribution on the weights. The distribution is centered at the origin and has variance $1/\lambda$. For multilayer networks, it is hard to find a good theoretical justification for this prior, but Hinton (1987) justifies it empirically by showing that it greatly improves generalization on a very difficult task. More recently, Mackay (1991) has shown that even better generalization can be achieved by using different values of $\lambda$ for the weights in different layers.

## 2    A more complex measure of network complexity

If we wish to eliminate small weights without forcing large weights away from the values they need to model the data, we can use a prior which is a mixture of a narrow ($n$) and a broad ($b$) gaussian, both centered at zero.

$$p(w) = \pi_n \frac{1}{\sqrt{2\pi}\sigma_n} e^{-\frac{w^2}{2\sigma_n^2}} + \pi_b \frac{1}{\sqrt{2\pi}\sigma_b} e^{-\frac{w^2}{2\sigma_b^2}} \tag{2}$$

where $\pi_n$ and $\pi_b$ are the mixing proportions of the two gaussians and are therefore constrained to sum to 1.

Assuming that the weight values were generated from a gaussian mixture, the conditional probability that a particular weight, $w_i$, was generated by a particular gaussian, $j$, is called the *responsibility* of that gaussian for the weight and is:

$$r_j(w_i) = \frac{\pi_j p_j(w_i)}{\sum_k \pi_k p_k(w_i)} \tag{3}$$

where $p_j(w_i)$ is the probability density of $w_i$ under gaussian $j$.

When the mixing proportions of the two gaussians are comparable, the narrow gaussian gets most of the responsibility for a small weight. Adopting the Bayesian perspective, the cost of a weight under the narrow gaussian is proportional to $w^2/2\sigma_n^2$. As long as $\sigma_n$ is quite small there will be strong pressure to reduce the magnitude

of small weights even further. Conversely, the broad gaussian takes most of the responsibility for large weight values, so there is much less pressure to reduce them. In the limiting case when the broad gaussian becomes a uniform distribution, there is almost no pressure to reduce very large weights because they are almost certainly generated by the uniform distribution. A complexity term very similar to this limiting case is used in the "weight elimination" technique of (Weigend, Huberman and Rumelhart, 1990) to improve generalization for a time series prediction task. [2]

## 3   Adaptive Gaussian Mixtures and Soft Weight-Sharing

A mixture of a narrow, zero-mean gaussian with a broad gaussian or a uniform allows us to favor networks with many near-zero weights, and this improves generalization on many tasks. But practical experience with hand-coded weight constraints has also shown that great improvements can be achieved by constraining particular subsets of the weights to share the same value (Lang, Waibel and Hinton, 1990; Le Cun, 1989). Mixtures of zero-mean gaussians and uniforms cannot implement this type of symmetry constraint. If however, we use multiple gaussians and allow their means and variances to adapt as the network learns, we can implement a "soft" version of weight-sharing in which the learning algorithm decides for itself which weights should be tied together. (We may also allow the mixing proportions to adapt so that we are not assuming all sets of tied weights are the same size.)

The basic idea is that a gaussian which takes responsibility for a subset of the weights will squeeze those weights together since it can then have a lower variance and assign a higher probability density to each weight. If the gaussians all start with high variance, the initial division of weights into subsets will be very soft. As the variances shrink and the network learns, the decisions about how to group the weights into subsets are influenced by the task the network is learning to perform.

To make these intuitive ideas a bit more concrete, we may define a cost function of the general form given in (1):

$$C = \frac{K}{\sigma_y^2} \sum_c \frac{1}{2}(y_c - d_c)^2 - \sum_i \log \left( \sum_j \pi_j p_j(w_i) \right) \qquad (4)$$

where $\sigma_y^2$ is the variance of the squared error and each $p_j(w_i)$ is a gaussian density with mean $\mu_j$ and standard deviation $\sigma_j$. We optimize this function by adjusting the $w_i$ and the mixture parameters $\pi_j$, $\mu_j$, and $\sigma_j$, and $\sigma_y$.[3]

The partial derivative of $C$ with respect to each weight is the sum of the usual squared error derivative and a term due to the complexity cost for the weight:

$$\frac{\partial C}{\partial w_i} = \frac{K}{\sigma_y^2} \sum_c (y_c - d_c) \frac{\partial y_c}{\partial w_i} - \sum_j r_j(w_i) \frac{(\mu_j - w_i)}{\sigma_j^2} \qquad (5)$$

| Method | Train % Correct | Test % Correct |
|---|---|---|
| Vanilla Back Prop. | $100.0 \pm 0.0$ | $67.3 \pm 5.7$ |
| Cross Valid. | $98.8 \pm 1.1$ | $83.5 \pm 5.1$ |
| Weight Elimination | $100.0 \pm 0.0$ | $89.8 \pm 3.0$ |
| Soft-share - 5 Comp. | $100.0 \pm 0.0$ | $95.6 \pm 2.7$ |
| Soft-share - 10 Comp. | $100.0 \pm 0.0$ | $97.1 \pm 2.1$ |

Table 1: Summary of generalization performance of 5 different training techniques on the shift detection problem.

The derivative of the complexity cost term is simply a weighted sum of the difference between the weight value and the center of each of the gaussians. The weighting factors are the *responsibility* measures defined in equation 3 and if over time a single gaussian claims most of the responsibility for a particular weight the effect of the complexity cost term is simply to pull the weight towards the center of the responsible gaussian. The strength of this force is inversely proportional to the variance of the gaussian.

In the simulations described below, all of the parameters $(w_i, \mu_j, \sigma_j, \pi_j)$ are updated *simultaneously* using a conjugate gradient descent procedure. To prevent variances shrinking too fast or going negative we optimize $\log \sigma_j$ rather than $\sigma_j$. To ensure that the mixing proportions sum to 1 and are positive, we optimize $x_j$ where $\pi_j = exp(x_j)/\sum exp(x_k)$. For further details see (Nowlan and Hinton, 1992).

## 4   Simulation Results

We compared the generalization performance of soft weight-tying to other techniques on two different problems. The first problem, a 20 input, one output shift detection network, was chosen because it was binary problem for which solutions which generalize well exhibit a lot of repeated weight structure. The generalization performance of networks trained using the cost criterion given in equation 4 was compared to networks trained in three other ways: No cost term to penalize complexity; No explicit complexity cost term, but use of a validation set to terminate learning; Weight elimination (Weigend, Huberman and Rumelhart, 1990)[4]. The simulation results are summarized in Table 1.

The network had 20 input units, 10 hidden units, and a single output unit and contained 101 weights. The first 10 input units in this network were given a random binary pattern, and the second group of 10 input units were given the same pattern circularly shifted by 1 bit left or right. The desired output of the network was +1 for a left shift and −1 for a right shift. A data set of 2400 patterns was created by randomly generating a 10 bit string, and choosing with equal probability to shift the string left or right. The data set was divided into 100 training cases, 1000 validation cases, and 1300 test cases. The training set was deliberately chosen to be very small ($< 5\%$ of possible patterns) to explore the region in which complexity penalties should have the largest impact. Ten simulations were performed with each

[4]With a fixed value of $\lambda$ chosen by cross-validation.

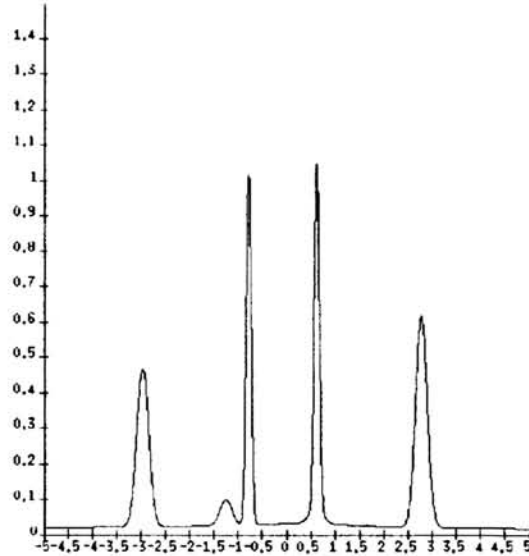

Figure 1: Final mixture probability density for a typical solution to the shift detection problem. Five of the components in the mixture can be seen as distinct bumps in the probability density. Of the remaining five components, two have been eliminated by having their mixing proportions go to zero and the other three are very broad and form the baseline offset of the density function.

method, starting from ten different initial weight sets (*i.e.* each method used the same ten initial weight configurations).

The final weight distributions discovered by the soft weight-tying technique are shown in Figure 1. There is no significant component with mean 0. The classical assumption that the network contains a large number of inessential weights which can be eliminated to improve generalization is not appropriate for this problem and network architecture. This may explain why the weight elimination model used by Weigend *et al* (Weigend, Huberman and Rumelhart, 1990) performs relatively poorly in this situation.

The second task chosen to evaluate the effectiveness of our complexity penalty was the prediction of the yearly sunspot average from the averages of previous years. This task has been well studied as a time-series prediction benchmark in the statistics literature (Priestley, 1991b; Priestley, 1991a) and has also been investigated by (Weigend, Huberman and Rumelhart, 1990) using a complexity penalty similar to the one discussed in section 2.

The network architecture used was identical to the one used in the study by Weigend *et al*. The network had 12 input units which represented the yearly average from the preceding 12 years, 8 hidden units, and a single linear output unit which represented the prediction for the average number of sunspots in the current year. Yearly sunspot data from 1700 to 1920 was used to train the network to perform this one-step prediction task, and the evaluation of the network was based on data from

| Method | Test arv |
|---|---|
| TAR | 0.097 |
| RBF | 0.092 |
| WRH | 0.086 |
| Soft-share - 3 Comp. | $0.077 \pm 0.0029$ |
| Soft-share - 8 Comp. | $0.072 \pm 0.0022$ |

Table 2: Summary of average relative variance of 5 different models on the one-step sunspot prediction problem.

1921 to 1955.[5] The evaluation of prediction performance used the *average relative variance* (*arv*) measure discussed in (Weigend, Huberman and Rumelhart, 1990).

Simulations were performed using the same conjugate gradient method used for the first problem. Complexity measures based on gaussian mixtures with 3 and 8 components were used and ten simulations were performed with each (using the same training data but different initial weight configurations). The results of these simulations are summarized in Table 2 along with the best result obtained by Weigend *et al* (Weigend, Huberman and Rumelhart, 1990) (*WRH*), the bilinear auto-regression model of Tong and Lim (Tong and Lim, 1980) (*TAR*)[6], and the multi-layer RBF network of He and Lapedes (He and Lapedes, 1991) (*RBF*). All figures represent the *arv* on the test set. For the mixture complexity models, this is the *average* over the ten simulations, plus or minus one standard deviation.

Since the results for the models other than the mixture complexity trained networks are based on a single simulation it is difficult to assign statistical signifigance to the differences shown in Table 2. We may note however, that the difference between the 3 and 8 component mixture complexity models is significant ($p > 0.95$) and the differences between the 8 component model and the other models are much larger.

Figure 2 shows an 8 component mixture model of the final weight distribution. It is quite unlike the distribution in Figure 1 and is actually quite close to a mixture of two zero-mean gaussians, one broad and one narrow. This may explain why weight elimination works quite well for this task.

Weigend *et al* point out that for time series prediction tasks such as the sunspot task a much more interesting measure of performance is the ability of the model to predict more than one time step into the future. One way to approach the multi-step prediction problem is to use *iterated single-step prediction*. In this method, the predicted output is fed back as input for the next prediction and all other input units have their values shifted back one unit. Thus the input typically consists of a combination of actual and predicted values. When predicting more than one step into the future, the prediction error depends both on how many steps into the future one is predicting (*I*) and on what point in the time series the prediction began. An appropriate error measure for iterated prediction is the *average relative I-times iterated prediction variance* (Weigend, Huberman and Rumelhart, 1990)

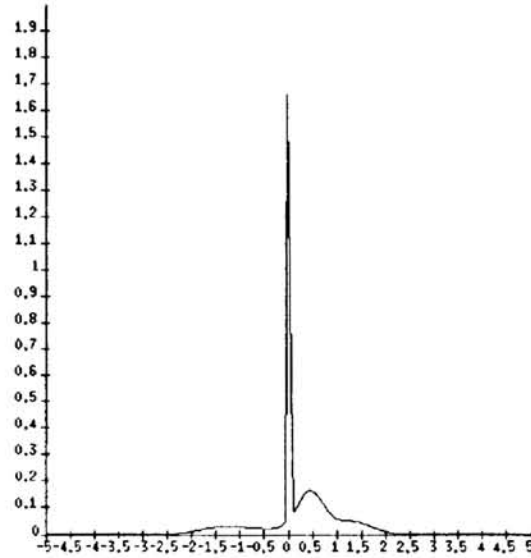

Figure 2: Typical final mixture probability density for the sunspot prediction problem with a model containing 8 mixture components.

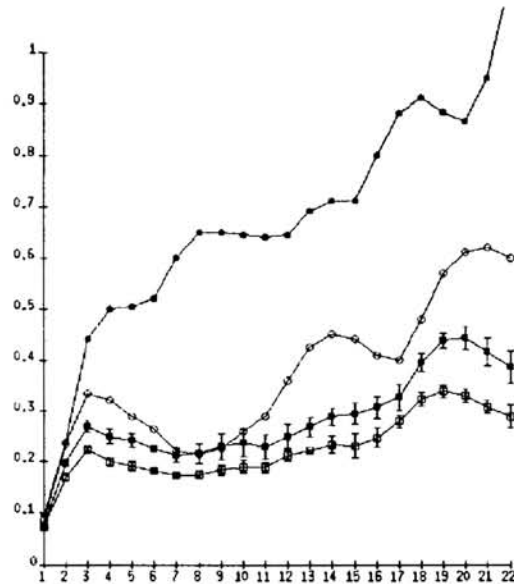

Figure 3: Average relative 1-times iterated prediction variance versus number of prediction iterations for the sunspot time series from 1921 to 1955. Closed circles represent the $TAR$ model, open circles the $WRH$ model, closed squares the 3 component complexity model, and open squares the 8 component complexity model. Ten different sets of initial weights were used for the 3 and 8 component complexity models and one standard deviation error bars are shown.

which averages predictions $I$ steps into the future over all possible starting points. Using this measure, the performance of various models is shown in Figure 3.

## 5 Summary

The simulations we have described provide evidence that the use of a more flexible model for the distribution of weights in a network can lead to better generalization performance than weight decay, weight elimination, or techniques that control the learning time. The flexibility of our model is clearly demonstrated in the very different final weight distributions discovered for the two different problems investigated in this paper. The ability to automatically adapt to individual problems suggests that the method should have broad applicability.

**Acknowledgements**

This research was funded by the Ontario ITRC, the Canadian NSERC and the Howard Hughes Medical Institute. Hinton is the Noranda fellow of the Canadian Institute for Advanced Research.

## Footnotes

[1]R. Szeliski, personal communication, 1985.

[2]See (Nowlan, 1991) for a precise description of the relationship between mixture models and the model used by (Weigend, Huberman and Rumelhart, 1990).

[3]$1/\sigma_y^2$ may be thought of as playing the same role as $\lambda$ in equation 1 in determining a trade-off between the misfit and complexity costs. $K$ is a normalizing factor based on a gaussian error model.

[5]The authors thank Andreas Weigend for providing his version of this data.

[6]This was the model favored by Priestly (Priestley, 1991a) in a recent evaluation of classical statistical approaches to this task.

## References

Baum, E. B. and Haussler, D. (1989). What size net gives valid generalization? *Neural Computation*, 1:151–160.

He, X. and Lapedes, A. (1991). Nonlinear modelling and prediction by successive approximation using Radial Basis Functions. Technical Report LA-UR-91-1375, Los Alamos National Laboratory.

Hinton, G. E. (1987). Learning translation invariant recognition in a massively parallel network. In *Proc. Conf. Parallel Architectures and Languages Europe*, Eindhoven.

Lang, K. J., Waibel, A. H., and Hinton, G. E. (1990). A time-delay neural network architecture for isolated word recognition. *Neural Networks*, 3:23–43.

Le Cun, Y. (1989). Generalization and network design strategies. Technical Report CRG-TR-89-4, University of Toronto.

MacKay, D. J. C. (1991). *Bayesian Modelling and Neural Networks*. PhD thesis, Computation and Neural Systems, California Institute of Technology, Pasadena, CA.

Nowlan, S. J. (1991). *Soft Competitive Adaptation: Neural Network Learning Algorithms based on Fitting Statistical Mixtures*. PhD thesis, School of Computer Science, Carnegie Mellon University, Pittsburgh, PA.

Nowlan, S. J. and Hinton, G. E. (1992). Simplifying neural networks by soft weight-sharing. *Neural Computation*. In press.

Plaut, D. C., Nowlan, S. J., and Hinton, G. E. (1986). Experiments on learning by back-propagation. Technical Report CMU-CS-86-126, Carnegie-Mellon University, Pittsburgh PA 15213.

Priestley, M. B. (1991a). *Non-linear and Non-stationary Time Series Analysis*. Academic Press.

Priestley, M. B. (1991b). *Spectral Analysis and Time Series*. Academic Press.

Tong, H. and Lim, K. S. (1980). Threshold autoregression, limit cycles, and cyclical data. *Journal Royal Statistical Society B*, 42.

Weigend, A. S., Huberman, B. A., and Rumelhart, D. E. (1990). Predicting the future: A connectionist approach. *International Journal of Neural Systems*, 1.